# Boosting the Performance of RBF Networks with Dynamic Decay Adjustment

**Michael R. Berthold**
Forschungszentrum Informatik
Gruppe ACID (Prof. D. Schmid)
Haid–und–Neu–Strasse 10—14
76131 Karlsruhe, Germany
eMail: berthold@fzi.de

**Jay Diamond**
Intel Corporation
2200 Mission College Blvd.
Santa Clara, CA, USA
95052 MS:SC9-15
eMail: jdiamond@mipos3.intel.com

## Abstract

Radial Basis Function (RBF) Networks, also known as networks of locally–tuned processing units (see [6]) are well known for their ease of use. Most algorithms used to train these types of networks, however, require a fixed architecture, in which the number of units in the hidden layer must be determined before training starts. The RCE training algorithm, introduced by Reilly, Cooper and Elbaum (see [8]), and its probabilistic extension, the P–RCE algorithm, take advantage of a growing structure in which hidden units are only introduced when necessary. The nature of these algorithms allows training to reach stability much faster than is the case for gradient–descent based methods. Unfortunately P–RCE networks do not adjust the standard deviation of their prototypes individually, using only one global value for this parameter.

This paper introduces the Dynamic Decay Adjustment (DDA) algorithm which utilizes the constructive nature of the P–RCE algorithm together with independent adaptation of each prototype's decay factor. In addition, this radial adjustment is class dependent and distinguishes between different neighbours. It is shown that networks trained with the presented algorithm perform substantially better than common RBF networks.

## 1   Introduction

Moody and Darken proposed Networks with locally–tuned processing units, which are also known as *Radial Basis Functions* (RBFs, see [6]). Networks of this type have a single layer of units with a selective response for some range of the input variables. Each unit has an overall response function, possibly a Gaussian:

$$R_i(\vec{x}) = exp(-\frac{||\vec{x} - \vec{r}_i||^2}{\sigma_i^2}) \tag{1}$$

Here $\vec{x}$ is the input to the network, $\vec{r}_i$ denotes the center of the $i$–th RBF and $\sigma_i$ determines its standard deviation. The second layer computes the output function for each class as follows:

$$f(\vec{x}) = \sum_{i=1}^{m} A_i * R_i(\vec{x}) \tag{2}$$

with $m$ indicating the number of RBFs and $A_i$ being the weight for each RBF. Moody and Darken propose a hybrid training, a combination of unsupervised clustering for the centers and radii of the RBFs and supervised training of the weights. Unfortunately their algorithm requires a fixed network topology, which means that the number of RBFs must be determined in advance. The same problem applies to the Generalized Radial Basis Functions (GRBF), proposed in [12]. Here a gradient descent technique is used to implement a supervised training of the center locations, which has the disadvantage of long training times.

In contrast RCE (*Restricted Coulomb Energy*) Networks construct their architecture dynamically during training (see [7] for an overview). This algorithm was inspired by systems of charged particles in a three–dimensional space and is analogous to the Liapunov equation:

$$\xi = -\frac{1}{L} \sum_{i=1}^{m} \frac{Q_i}{||\vec{x} - \vec{r}_i||_2^L} \tag{3}$$

where $\xi$ is the electrostatic potential induced by fixed particles with charges $-Q_i$ and locations $\vec{r}_i$. One variation of this type of networks is the so called P–RCE network, which attempts to classify data using a probabilistic distribution derived from the training set. The underlying training algorithm for P–RCE is identical to RCE training with gaussian activation functions used in the forward pass to resemble a Probabilistic Neural Network (PNN [10]). PNNs are not suitable for large databases because they commit one new prototype for each training pattern they encounter, effectively becoming a referential memory scheme. In contrast, the P–RCE algorithm introduces a new prototype only when necessary. This occurs when the prototype of a conflicting class misclassifies the new pattern during the training phase. The probabilistic extension is modelled by incrementing the a–priori rate of occurrence for prototypes of the same class as the input vector, therefore weights are only connecting RBFs and an output node of the same class. The recall phase of the P–RCE network is similar to RBFs, except that it uses one global radius for all prototypes and scales each gaussian by the a-priori rate of occurrence:

$$act_c(\vec{x}) = \sum_{i=1}^{m_c} A_i^c * exp(-\frac{||\vec{x} - \vec{r}_i^c||^2}{R^2}) \tag{4}$$

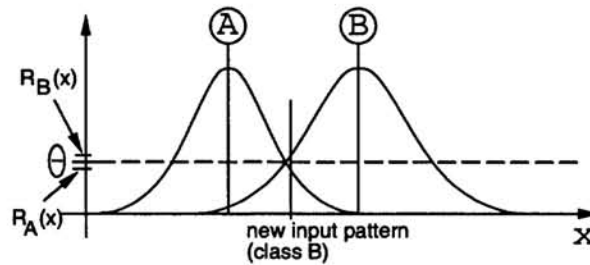

Figure 1: This picture shows how a new pattern results in a slightly higher activity for a prototype of the right class than for the conflicting prototype. Using only one threshold, no new prototype would be introduced in this case.

where $c$ denotes the class for which the activation is computed, $m_c$ is the number of prototypes for class $c$, and $R$ is the constant radius of the gaussian activation functions. The global radius of this method and the inability to recognize areas of conflict, leads to confusion in some areas of the feature space, and therefore non–optimal recognition performance.

The Dynamic Decay Adjustment (DDA) algorithm presented in this paper was developed to solve the inherent problems associated with these methods. The constructive part of the P–RCE algorithm is used to build a network with an appropriate number of RBF units, for which the decay factor is computed based on information about neighbours. This technique increases the recognition accuracy in areas of conflict.

The following sections explain the algorithm, compare it with others, and examine some simulation results.

## 2  The Algorithm

Since the P–RCE training algorithm already uses an independent *area of influence* for each RBF, it is relatively straightforward to extract an individual radius. This results, however, in the problem illustrated in figure 1. The new pattern $\vec{p}$ of class $B$ is properly covered by the right prototype of the same class. However, the left prototype of conflicting class $A$ results in almost the same activation and this leads to a very low confidence when the network must classify the pattern $\vec{p}$.

To solve this dilemma, two different radii, or thresholds[1] are introduced: a so–called *positive threshold* $(\theta^+)$, which must be overtaken by an activation of a prototype of the same class so that no new prototype is added, and a *negative threshold* $(\theta^-)$, which is the upper limit for the activation of conflicting classes. Figure 2 shows an example in which the new pattern correctly results in activations above the positive threshold for the correct class $B$ and below the negative threshold for conflicting class $A$. This results in better classification–confidence in areas where training

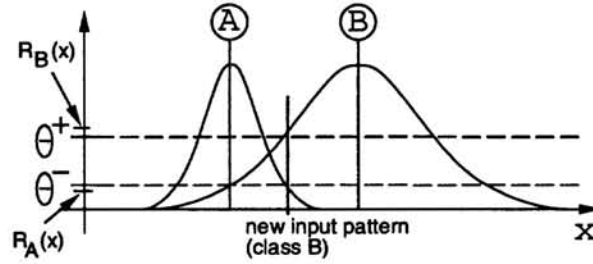

Figure 2: The proposed algorithm distinguishes between prototypes of correct and conflicting classes and uses different thresholds. Here the level of confidence is higher for the correct classification of the new pattern.

patterns did not result in new prototypes. The network is required to hold the following two equations for every pattern $\vec{x}$ of class $c$ from the training data:

$$\exists i : R_i^c(\vec{x}) \geq \theta^+ \tag{5}$$

$$\forall k \neq c, 1 \leq j \leq m_k : R_j^k(\vec{x}) < \theta^- \tag{6}$$

The algorithm to construct a classifier can be extracted partly from the RCE algorithm. The following pseudo code shows what the training for one new pattern $\vec{x}$ of class $c$ looks like:

```
// reset weights:
FORALL prototypes p_i^k DO
    A_i^k = 0.0
ENDFOR
// train one complete epoch
FORALL training pattern (x, c) DO:
    IF ∃p_i^c : R_i^c(x) ≥ θ^+ THEN
        A_i^c + = 1.0
    ELSE
        // "commit": introduce new prototype
        add new prototype p_{m_c+1}^c with:
        r_{m_c+1}^c = x
        σ_{m_c+1}^c =    max      {σ : R_{m_c+1}^c(r_j^*) < θ^-}
                    k≠c∧1≤j≤m_k
        A_{m_c+1}^c = 1.0
        m_c + = 1
    ENDIF
    // "shrink": adjust conflicting prototypes
    FORALL k ≠ c, 1 ≤ j ≤ m_k DO
        σ_j^k = max{σ : R_j^k(x) < θ^-}
ENDFOR
```

First, all weights are set to zero because otherwise they would accumulate duplicate information about training patterns. Next all training patterns are presented to the

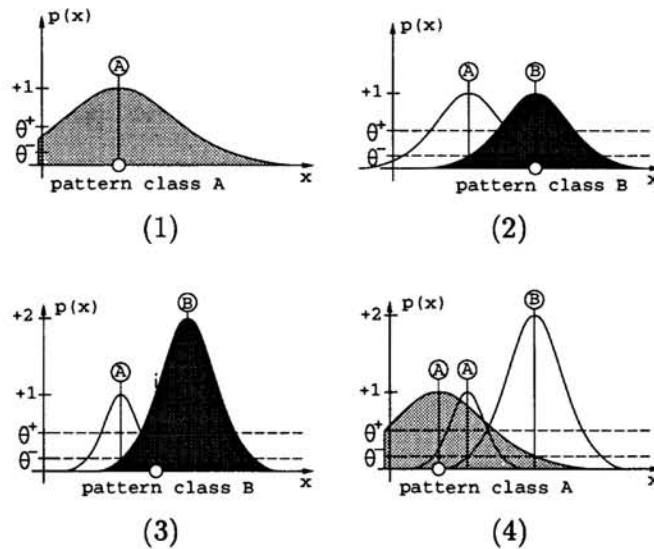

Figure 3: An example of the DDA–algorithm: (1) a pattern of class $A$ is encountered and a new RBF is created; (2) a training pattern of class $B$ leads to a new prototype for class $B$ and shrinks the radius of the existing RBF of class $A$; (3) another pattern of class $B$ is classified correctly and shrinks again the prototype of class $A$; (4) a new pattern of class $A$ introduces another prototype of that class.

network. If the new pattern is classified correctly, the weight of the closest prototype is increased; otherwise a new protoype is introduced with the new pattern defining its center. The last step of the algorithm shrinks all prototypes of conflicting classes if their activations are too high for this specific pattern.

Running this algorithm over the training data until no further changes are required ensures that equations (5) and (6) hold.

The choice of the two new parameters, $\theta^+$ and $\theta^-$ are not as critical as it would initially appear[2]. For all of the experiments reported, the settings $\theta^+ = 0.4$ and $\theta^- = 0.1$ were used, and no major correlations of the results to these values were noted. Note that when choosing $\theta^+ = \theta^-$ one ends up with an algorithm having the problem mentioned in figure 1.

Figure 3 shows an example that illustrates the first few training steps of the DDA–algorithm.

## 3 Results

Several well-known databases were chosen to evaluate this algorithm (some can be found in the CMU Neural Network Benchmark Databases (see [13])). The DDA–

algorithm was compared against PNN, RCE and P–RCE as well as a classic Multi Layer Perceptron which was trained using a modified Backpropagation algorithm (Rprop, see [9]). The number of hidden nodes of the MLP was optimized manually. In addition an RBF–network with a fixed number of hidden nodes was trained using unsupervised clustering for the center positions and a gradient descent to determine the weights (see [6] for more details). The number of hidden nodes was again optimized manually.

- Vowel Recognition: Speaker independent recognition of the eleven steady state vowels of *British* English using a specified training set of Linear Predictive Coding (LPC) derived log area ratios (see [3]) resulting in 10 inputs and 11 classes to distinguish. The training set consisted of 528 tokens, with 462 different tokens used to test the network.

| algorithm | performance | #units | #epochs |
|---|---|---|---|
| Nearest Neighbour | 56% | — | 1 |
| MLP (RPROP) | 57% | 5 | ~200 |
| PNN | 61% | 528 | — |
| RBF | 59% | 70 | ~100 |
| RCE | 27% | 125 | 3 |
| P–RCE | 59% | 125 | 3 |
| DDA–RBF | **65%** | 204 | 4 |

- Sonar Database: Discriminate between sonar signals bounced off a metal cylinder and those bounced off a roughly cylindrical rock (see [4] for more details). The data has 60 continuous inputs and is separated into two classes. For training and testing 104 samples each were used.

| algorithm | performance | #units | #epochs |
|---|---|---|---|
| MLP (RPROP) | 90.4% | 50 | ~250 |
| PNN | 91.3% | 104 | — |
| RBF | 90.7% | 80 | ~150 |
| RCE | 77.9% | 68 | 3 |
| P–RCE | 90.4% | 68 | 3 |
| DDA–RBF | **93.3%** | 68 | 3 |

- Two Spirals: This well–known problem is often used to demonstrate the generalization capability of a network (see [5]). The required task involves discriminating between two intertwined spirals. For this paper the spirals were changed slightly to make the problem more demanding. The original spirals radius declines linearly and can be correctly classified by RBF networks with one global radius. To demonstrate the ability of the DDA–algorithm to adjust the radii of each RBF individually, a quadratic decline was chosen for the radius of both spirals (see figure 4). The training set consisted of 194 points, and the spirals made three complete revolutions. Figure 4 shows both the results of an RBF Network trained with the DDA technique and the same problem solved with a Multi-Layer Perceptron (2–20–20–1) trained using a modified Error Back Propagation algorithm (Rprop, see [9]). Note that in both cases all training points are classified correctly.

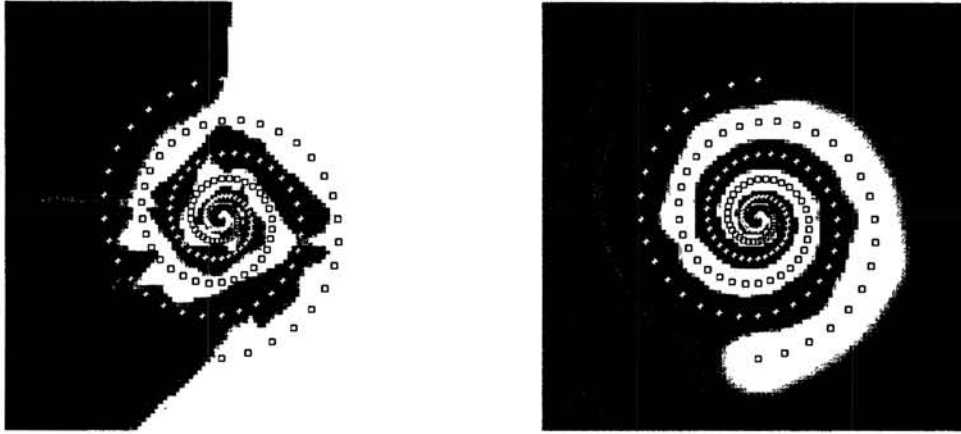

Figure 4: The (quadratic) "two spirals problem" solved by a MLP (left) using Error Back Propagation (after 40000 epochs) and an RBF network (right) trained with the proposed DDA–algorithm (after 4 epochs). Note that all training patterns (indicated by squares vs. crosses) are classified correctly.

In addition to these tasks, the BDG–database was used to compare the DDA algorithm to other approaches. This database was used by Waibel et al (see [11]) to introduce the Time Delay Neural Network (TDNN). Previously it has been shown that RBF networks perform equivalently (when using a similar architecture, [1], [2]) with the DDA technique used for training of the RBF units. The BDG task involves distinguishing the three stop consonants "B", "D" and "G". While 783 training sets were used, 749 data sets were used for testing. Each of these contains 15 frames of melscale coefficients, computed from a 10kHz, 12bit converted signal. The final frame frequency was 100Hz.

| algorithm | performance | #epochs |
|:---:|:---:|:---:|
| TDNN | 98.5% | ~50 |
| TDRBF (P–RCE) | 85.2% | 5 |
| TDRBF (DDA) | **98.3%** | 6 |

## 4   Conclusions

It has been shown that Radial Basis Function Networks can boost their performance by using the dynamic decay adjustment technique. The algorithm necessary to construct RBF networks based on the RCE method was described and a method to distinguish between conflicting and matching prototypes at the training phase was proposed. An increase in performance was noted, especially in areas of conflict, where standard (P–)RCE did not commit new prototypes.

Four different datasets were used to show the performance of the proposed DDA–algorithm. In three of the cases, RBF networks trained with dynamic decay adjustment outperformed known RBF training methods and MLPs. For the fourth task, the BDG–recognition dataset, the TDRBF was able to reach the same level

of performance as a TDNN.

In addition, the new algorithm trains very quickly. Fewer than 6 epochs were sufficient to reach stability for all problems presented.

### Acknowledgements

Thanks go to our supervisors Prof. D. Schmid and Mark Holler for their support and the opportunity to work on this project.

## Footnotes

[1] The conversion from the threshold to the radius is straightforward as long as the activation function is invertible.

[2]Theoretically one would expect the dimensionality of the input–space to play a major role for the choice of those parameters

## References

[1] M. R. Berthold: "A Time Delay Radial Basis Function Network for Phoneme Recognition" in Proc. of the IEEE International Conference on Neural Networks, 7, p.4470–4473, 1994.

[2] M. R. Berthold: "The TDRBF: A Shift Invariant Radial Basis Function Network" in Proc. of the Irish Neural Network Conference, p.7–12, 1994.

[3] D. Deterding: "Speaker Normalization for Automatic Speech Recognition", PhD Thesis, University of Cambridge, 1989.

[4] R. Gorman, T. Sejnowski: "Analysis of Hidden Units in a Layered Network Trained to Classify Sonar Targets" in Neural Networks 1, pp.75.

[5] K. Lang, M. Witbrock: "Learning to Tell Two Spirals Apart", in Proc. of Connectionist Models Summer School, 1988.

[6] J. Moody, C.J. Darken: "Fast Learning in Networks of Locally–Tuned Processing Units" in Neural Computation 1, p.281–294, 1989.

[7] M.J. Hudak: "RCE Classifiers: Theory and Practice" in Cybernetics and Systems 23, p.483–515, 1992.

[8] D.L. Reilly, L.N. Cooper, C. Elbaum: "A Neural Model for Category Learning" in Biol. Cybernet. 45, p.35–41, 1982.

[9] M. Riedmiller, H. Braun: "A Direct Adaptive Method for Faster Backpropagation Learning: The Rprop Algorithm" in Proc. of the IEEE International Conference on Neural Networks, 1, p.586–591, 1993.

[10] D.F. Specht: "Probabilistic Neural Networks" in Neural Networks 3, p.109–118, 1990.

[11] A. Waibel, T. Hanazawa, G. Hinton, K. Shikano, K. Lang: "Phoneme Recognition Using Time–Delay Neural Networks" in IEEE Trans. in Acoustics, Speech and Signal Processing Vol. 37, No. 3, 1989.

[12] D. Wettschereck, T. Dietterich: "Improving the Performance of Radial Basis Function Networks by Learning Center Locations" in Advances in Neural Information Processing Systems 4, p.1133–1140, 1991.

[13] S. Fahlman, M. White: "The Carnegie Mellon University Collection of Neural Net Benchmarks" from ftp.cs.cmu.edu in /afs/cs/project/connect/bench.